# Temporally Asymmetric Hebbian Learning, Spike Timing and Neuronal Response Variability

**L.F. Abbott and Sen Song**
Volen Center and Department of Biology
Brandeis University
Waltham MA 02454

## Abstract

Recent experimental data indicate that the strengthening or weakening of synaptic connections between neurons depends on the relative timing of pre- and postsynaptic action potentials. A Hebbian synaptic modification rule based on these data leads to a stable state in which the excitatory and inhibitory inputs to a neuron are balanced, producing an irregular pattern of firing. It has been proposed that neurons *in vivo* operate in such a mode.

## 1 Introduction

Hebbian modification of network interconnections plays a central role in the study of learning in neural networks (Rumelhart and McClelland, 1986; Hertz *et al.*, 1991). Most work on Hebbian learning involves network models in which the activities of the individual units are represented by continuous variables. A Hebbian learning rule, in this context, is specified by describing how network weights change as a function of the activities of the units that transmit and receive signals across a given network connection. While analyses of Hebbian learning along these lines have provided important results, direct application of these ideas to neuroscience is hindered by the fact that real neurons cannot be adequately described by continuous activity variables such as firing rates. Instead, the inputs and outputs of neurons are sequences of action potentials or spikes. All the information conveyed by one neuron to another over any appreciable distance is carried by the temporal patterns of action potential sequences. Rules by which synaptic connections between real neurons are modified in a Hebbian manner should properly be expressed as functions of the relative timing of the action potentials fired by the input (presynaptic) and output (postsynaptic) neurons. Until recently, little information has been available about the exact dependence of synaptic modification on pre- and postsynaptic spike timing (see however, Levy and Steward, 1983; Gustafsson *et al.*, 1987). New experimental results (Markram *et al.*, 1997; Bell *et al.*, 1997; Debanne *et al.*, 1998; Zhang *et al.*, 1998; Bi and Poo, 1999) have changed

this situation dramatically, and these allow us to study Hebbian learning in a manner that is much more realistic and relevant to biological neural networks. The results may find application in artificial neural networks as well.

## 2  Temporally Asymmetric LTP and LTD

The biological substrate for Hebbian learning in neuroscience is provided by long-term potentiation (LTP) and long-term depression (LTD) of the synaptic connections between neurons (see for example, Malenka and Nicoll, 1993). LTP is a long-lasting strengthening of synaptic efficacy associated with paired pre- and postsynaptic activity. LTD is a long-lasting weakening of synaptic strength. In recent experiments on neocortical slices (Markram *et al.*, 1997), hippocampal cells in culture (Bi and Poo, 1999), and *in vivo* studies of tadpole tectum (Zhang *et al.*, 1998), induction of LTP required that presynaptic action potentials preceded postsynaptic firing by no more than about 20 ms. Maximal LTP occurred when presynaptic spikes preceded postsynaptic action potentials by less than a few milliseconds. If presynaptic spikes followed postsynaptic action potentials, long-term depression rather than potentiation resulted. These results are summarized schematically in Figure 1.

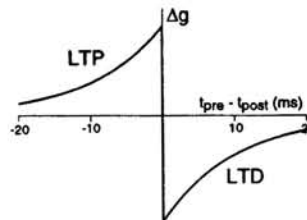

Figure 1: A model of the change in synaptic strength $\Delta g$ produced by paired pre- and postsynaptic spikes occurring at times $t_{pre}$ and $t_{post}$ respectively. Positive changes correspond to LTP and negative to LTD. There is an abrupt transition at $t_{pre} - t_{post} = 0$. The units for $\Delta g$ are arbitrary in this figure, but data indicate a maximum change of approximately 0.5 % per spike pair.

The curve in Figure 1 is a caricature used to model the weight changes arising from pairings of pre- and postsynaptic action potentials separated by various intervals of time. This curve resembles the data from all three preparations discussed above, but a couple of assumptions have been made in its construction. The data indicate that there is a rapid transition from LTP to LTD depending on whether the time difference between pre- and postsynaptic spiking is positive or negative, but the existing data cannot resolve exactly what happens at the transition point. We have assumed that there is a discontinuous jump from LTP to LTD at this point. In addition, we assume that the area under the LTP side of the curve is slightly less than the area under the LTD side. In Figure 1, this difference is imposed by making the magnitude of LTD slightly greater than the magnitude of LTP, while both sides of the curve have equal exponential fall-offs away from zero time difference. Alternately, we could have given the LTD side a slower exponential fall-off and equal amplitude. The data do not support either assumption unambiguously, nor do they indicate which area is larger. The assumption that the area under the LTD side of the curve is larger than that under the LTP side is critical if the resulting synaptic modification rule is to be stable against uncontrolled growth of synaptic strengths.

Hebb (1949) postulated that a synapse should be strengthened when the presynaptic neuron is frequently involved in making the postsynaptic neuron fire an action potential. Causality is an important element in Hebb's statement; synaptic potentiation should occur only if there is a causal relationship between the pre- and postsynaptic spiking. The LTP/LTD rule summarized in Figure 1 imposes causality through a tight timing requirement. The narrow

windows for LTP and LTD seen in the data, and the abrupt transition from potentiation to depression near zero separation between pre- and postsynaptic spike times impose a strict causality condition for LTP induction.

## 3 Response Variability

What are the implications of the synaptic modification rule summarized in Figure 1? To address this question, we introduce another topic that has been discussed extensively within the computational neuroscience community in recent years, the origin of response variability (Softky and Koch, 1992 & 1994; Shadlen and Newsome, 1994 & 1998; Tsodyks and Sejnowski, 1995; Amit and Brunel, 1997; Troyer and Miller, 1997a & b; Bugmann *et al.*, 1997; van Vreeswijk and Sompolinsky, 1996 & 1998). Neurons can respond to multiple synaptic inputs in two different modes of operation. Figure 2 shows membrane potentials of a model neuron receiving 1000 excitatory and 200 inhibitory synaptic inputs. Each input consists of an independent Poisson spike train driving a synaptic conductance. The integrate-and-fire model neuron used in this example integrates these synaptic conductances as a simple capacitor-resistor circuit. To generate action potentials in this model, we monitor the membrane potential and compare it to a threshold voltage. Whenever the membrane potential reaches the threshold an action potential is "pasted" onto the membrane potential trace and the membrane potential is reset to a prescribed value.

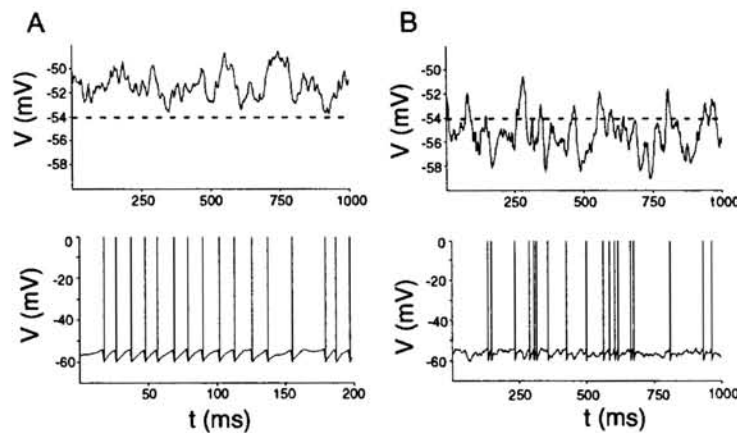

Figure 2: Regular and irregular firing modes of a model integrate-and-fire neuron. Upper panels show the model with action potentials deactivated, and the dashed lines show the action potential threshold. The lower figures show the model with action potentials activated. A) In the regular firing mode, the average membrane potential without spikes is above threshold and the firing pattern is fast and regular (note the different time scale in the lower panel). B) In the irregular firing mode, the average membrane potential without spikes is below threshold and the firing pattern is slower and irregular.

Figures 2A and 2B illustrate the two modes of operation. The upper panels of Figure 2 show the membrane potential with the action potential generation mechanism of the model turned off, and the lower panels show the membrane potential and spike sequences that result when the action potential generation is turned on. In Figure 2A, the effect of the excitatory inputs is strong enough relative to that of the inhibitory inputs so that the average membrane potential, when action potential generation is blocked, is above the spike threshold of the model. When the action potential mechanism is turned back on (lower panel of Figure 2A), this produces a fairly regular pattern of action potentials at a relatively high rate. The total synaptic input attempts to charge the neuron above the threshold, but every time the potential reaches the threshold it gets reset and starts charging again. In this regular

firing mode of operation, the timing of the action potentials is determined primarily by the charging rate of the cell, which is controlled by its membrane time constant. Since this does not vary as a function of time, the firing pattern is regular despite the fact that the synaptic input is varying.

Figure 2B shows the other mode of operation that produces an irregular firing pattern. In the irregular firing mode, the average membrane is more hyperpolarized than the threshold for action potential generation (upper panel of Figure 2B). In this mode, action potentials are only generated when there is a fluctuation in the total synaptic current strong enough to make the membrane potential cross the threshold. This results in slower and more irregular firing (lower panel of Figure 2B). The irregular firing mode has a number of interesting features (Shadlen and Newsome, 1994 & 1998; Tsodyks and Sejnowski, 1995; Amit and Brunel, 1997; Troyer and Miller, 1997a & b; Bugmann *et al.*, 1997; van Vreeswijk and Sompolinsky, 1996 & 1998). First, it generates irregular firing patterns that are far closer to the firing patterns seen *in vivo* than the patterns produced in the regular firing mode. Second, responses to changes in the synaptic input are much more rapid in this mode, being limited only by the synaptic rise time rather than the membrane time constant. Finally, the timing of action potentials in the irregular firing mode is related to the timing of fluctuations in the synaptic input rather than being determined primarily by the membrane time constant of the cell.

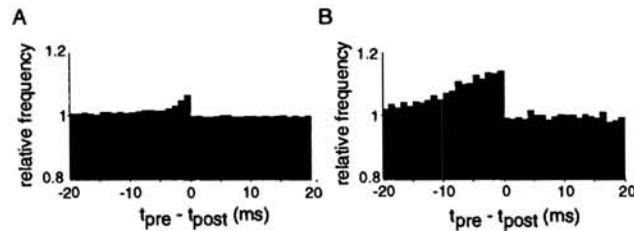

Figure 3: Histograms indicating the relative probability of finding pre- and postsynaptic spikes separated by the indicated time interval. A) Regular firing mode. The probability is essentially flat and at the chance level of one. B) Irregular firing mode. There is an excess of presynaptic spike shortly before a postsynaptic spike.

An important difference between the regular and irregular firing modes is illustrated in the cross-correlograms shown in Figure 3 (Troyer and Miller, 1997b; Bugmann et al. 1997). These indicate the probability that an action potential fired by the postsynaptic neuron is preceded or followed by an presynaptic spike separated by various intervals. The histogram has been normalized so its value for pairings that are due solely to chance is one. The histogram when the model is in the regular firing mode (Figure 3A) takes a value close to one for almost all input-output spike time differences. This is a reflection of the fact that the timing of individual action potentials in the regular firing mode is relatively independent of the timing of the presynaptic inputs. In contrast, the histogram for a model neuron in the irregular firing mode (Figure 3B) shows a much larger excess of presynaptic spikes occurring shortly before the postsynaptic neuron fires. This excess reflects the fluctuations in the total synaptic input that push the membrane potential up to the threshold and produce a spike in the irregular firing mode. It indicates that, in this mode, there is a tight temporal correlation between the timing of such fluctuations and output spikes.

For a neuron to operate in the irregular firing mode, there must be an appropriate balance between the strength of its excitatory and inhibitory inputs. The excitatory input must be weak enough, relative to the inhibitory input, so that the average membrane potential in the absence of spikes is below the action potential threshold to avoid regular firing. However, excitatory input must be sufficiently strong to keep the average potential close enough to

the threshold so that fluctuations can reach it and cause the cell to fire. How is this balance achieved?

## 4 Asymmetric LTP/LTD Leads to an Irregular Firing State

A comparison of the LTP/LTD synaptic modification rule illustrated in Figure 1, and the presynaptic/postsynaptic timing histogram shown in Figure 3, reveals that a temporally asymmetric synaptic modification rule based on the curve in Figure 1 can automatically generate the balance of excitation and inhibition needed to produce an irregular firing state. Suppose that we start a neuron model in a regular firing mode by giving it relatively strong excitatory synaptic strengths. We then apply the LTP/LTD rule of Figure 1 to the excitatory synapse while holding the inhibitory synapse at constant values. Recall that Figure 1 has been adjusted so that the area under the LTD part of the curve is greater than that under the LTP part. This means that if there is an equal probability of a presynaptic spike to either precede or follow a postsynaptic spike the net effect will be a weakening of the excitatory synapses. This is exactly what happens in the regular firing mode, where the relationship between the timing of pre- and postsynaptic spikes is approximately random (Figure 3A). As the LTP/LTD rule weakens the excitatory synapses, the average membrane potential drops and the neuron enters the irregular firing mode. In the irregular firing mode, there is a higher probability for a presynaptic spike to precede than to follow a postsynaptic spike (Figure 3B). This compensates for the fact that the rule we use produces more LTD than LTP. Equilibrium will be reached when the asymmetry of the LTP/LTD modification curve of Figure 1 is matched by the asymmetry of the presynaptic/postsynaptic timing histogram of Figure 3B. The equilibrium state corresponds to a balanced, irregular firing mode of operation, and it is automatically produced by the temporally asymmetric learning rule.

Figure 4A shows a transition from a regular to an irregular firing state mediated by the temporally asymmetric LTP/LTD modification rule. The irregularity of the postsynaptic spike train has been quantified by plotting the coefficient of variation (CV), the standard deviation over the mean of the interspike intervals, of the model neuron as a function of time. Initially, the neuron was in a regular firing state with a low CV value. After the synaptic modification rule reached an equilibrium state, the CV took a value near one indicating that the neuron has been transformed into an irregular firing mode. The solid curve in Figure 4B shows that temporally asymmetric LTP/LTD can robustly generate irregular output firing for a wide range of input firing rates.

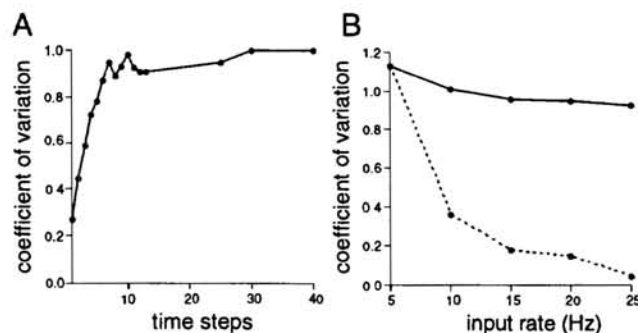

Figure 4: Coefficient of variation (CV) of the output spike train of the model neuron. A) Transition from a regular to an irregular firing state as temporally asymmetric LTP/LTD modifies synaptic strengths. The units of time in this plot are arbitrary because they depend on the magnitude of LTP and LTD used in the model. B) Equilibrium CV values as a function of the firing rates of excitatory inputs to the model neuron. The solid curve gives the results when temporally asymmetric LTP/LTD is active. The dashed curve shows the results if the synaptic strengths that arose for 5 Hz inputs are left unmodified.

# 5   Discussion

Temporally asymmetric LTP/LTD provides a Hebbian-type learning rule with interesting properties (Kempter *et al.*, 1998). In the past, temporally asymmetric Hebbian learning rules have been studied and applied to problems of temporal sequence generation (Manai and Levy, 1993), navigation (Blum and Abbott, 1996; Gerstner and Abbott, 1997), motor learning (Abbott and Blum, 1996), and detection of spike synchrony (Gerstner *et al.*, 1996). In these studies, two different LTP/LTD window sizes were assumed: either of order 100 ms (Manai and Levy, 1993; Blum and Abbott, 1996; Gerstner and Abbott, 1997; (Abbott and Blum, 1996) or around 1 ms (Gerstner *et al.*, 1996). The new data (Markram *et al.*, 1997; Bell *et al.*, 1997; Zhang *et al.*, 1998; Bi and Poo, 1999) give a window size of order 10 ms. For a 1 ms window size, temporally asymmetric LTP/LTD is sensitive to precise spike timing. When the window size is of order 100 ms, changes in stimuli or motor actions on a behavioral level become relevant for LTP and LTD. A window size of 10 ms, as supported by the recent data, suggests that LTP and LTD are sensitive to firing correlations relevant to neuronal circuitry, such as input-output correlations, which vary over this time scale.

Temporally asymmetric LTP/LTD has some interesting properties that distinguish it from Hebbian learning rules based on correlations or covariances in pre- and postsynaptic rates. We have found that the rule used here is not sensitive to input firing rates or to variability in input rates. If we split the excitatory inputs of the model into two groups and give these two input sets different rates, we see no difference in the distribution of synaptic strengths arising from the learning rule. Similarly, if one group is given a steady firing rate and the other group has firing rates that vary in time, no difference in synaptic strengths is apparent. The most effective way to induce LTP in a set of inputs is to synchronize some of their spikes. Inputs with synchronized spikes are slightly more effective at firing the neuron than unsynchronized spikes. This means that such inputs will preceded postsynaptic spikes more frequently and thus will get stronger. This suggests that spike synchrony may be a signal that marks a set of inputs for learning. Even when this synchrony has no particular functional effect, so that it has little impact on the firing pattern of the postsynaptic neuron, it can lead to dramatic shifts in synaptic strength. Thus, spike synchronization may be a mechanism for inducing LTP and LTD.

## Acknowledgments

Research supported by the National Science Foundation (DMS-9503261), the Sloan Center for Theoretical Neurobiology at Brandeis University, a Howard Hughes Predoctoral Fellowship, and the W.M. Keck Foundation.

## References

Abbott, LF & Blum, KI (1996) Functional significance of long-term potentiation for sequence learning and prediction. *Cerebral Cortex* **6**:406-416.

Amit, DJ & Brunel N (1997) Global spontaneous activity and local structured (learned) delay activity in cortex. *Cerebral Cortex* **7**:237-252.

Bell CC, Han VZ, Sugawara Y & Grant K (1997) Synaptic plasticity in a cerebellum-like structure depends on temporal order. *Nature* **387**:278-281.

Bi G-q & Poo M-m (1999) Activity-induced synaptic modifications in hippocampal culture: dependence on spike timing, synaptic strength and cell type. *J. Neurophysiol.* (in press).

Blum, KI & Abbott, LF (1996) A model of spatial map formation in the hippocampus of the rat. *Neural Comp.* **8**:85-93.

Bugmann, G, Christodoulou, C & and Taylor, JG (1997) Role of temporal integration and fluctuation detection in the highly irregular firing of a leaky integrator neuron model with partial reset. *Neural Compu.* **9**:985-1000.

Debanne D, Gahwiler BH, Thompson SM (1998) Long-term synaptic plasticity between pairs of individual CA3 pyramidal cells in rat hippocampal slices. *J. Physiol.* **507**:237-247.

Gerstner, W & Abbott, LF (1997) Learning navigational maps through potentiation and modulation of hippocampal place cells. *J. Computational Neurosci.* **4**:79-94.

Gerstner W, Kempter R, van Hemmen JL & Wagner, H (1996) A neural learning rule for sub-millisecond temporal coding. *Nature* **383**:76-78.

Gustafsson B, Wigstrom H, Abraham WC & Huang Y-Y (1987) Long-term potentiation in the hippocampus using depolarizing current pulses as the conditioning stimulus to single volley synaptic potentials. *J. Neurosci.* **7**:774-780.

Hebb, DO (1949) *The Organization of Behavior: A Neuropsychological Theory.* New York:Wiley.

Hertz, JA, Palmer, RG & Krogh, A (1991) *Introduction to the Theory of Neural Computation.* New York:Addison-Wesley.

Kempter R, Gerstner W & van Hemmen JL (1999) Hebbian learning and spiking neurons. (submitted).

Levy WB & Steward O (1983) Temporal contiguity requirements for long-term associative potentiation/depression in the hippocampus. *Neurosci.* **8**:791-797.

Malenka, RC & Nicoll, RA (1993) MBDA-receptor-dependent synaptic plasticity: Multiple forms and mechanisms. *Trends Neurosci.* **16**:521-527.

Minai, AA & Levy, WB (1993) Sequence learning in a single trial. *INNS World Congress on Neural Networks* **II**:505-508.

Markram H, Lubke J, Frotscher M & Sakmann B (1997) Regulation of synaptic efficacy by coincidence of postsynaptic APs and EPSPs. *Science* **275**:213-215.

Rumelhart, DE & McClelland, JL, editors (1986) *Parallel Distributed Processing: Explorations in the Microstructure of Cognition, Volumes I & II.* Cambridge, MA:MIT Press.

Shadlen, MN & Newsome, WT (1994) Noise, neural codes and cortical organization. *Current Opinion in Neurobiology* **4**:569-579.

Shadlen, MN & Newsome, WT (1998) The Variable Discharge of Cortical Neurons: Implications for Connectivity, Computation, and Information Coding. *Journal of Neuroscience* **18**:3870-3896.

Softky, WR & Koch, C (1992) Cortical cells should spike regularly but do not. *Neural Computation* **4**:643-646.

Softky, WR & Koch, C (1994) The highly irregular firing of cortical cells is inconsistent with temporal integration of random EPSPs. *Journal of Neuroscience* **13**:334-350.

Troyer, TW & Miller, KD (1997a) Physiological gain leads to high ISI variability in a simple model of a cortical regular spiking cell. *Neural Comp.* **9**:971-983.

Troyer, TW & Miller, KD (1997b) Integrate-and-fire neurons matched to physiological F-I curves yield high input sensitivity and wide dynamic range. *Computational Neuroscience, Trends in Research.* JM Boser, ed. New York:Plenum, pp. 197-201.

Tsodyks, M & Sejnowski, TJ (1995) Rapid switching in balanced cortical network models. *Network* **6**:1-14.

van Vreeswijk, C & Sompolinsky, H (1996) Chaos in neuronal networks with balanced excitatory and inhibitory activity. *Science* **274**:1724-1726.

van Vreeswijk, C & Sompolinsky, H (1998) Chaotic balanced state in a model of cortical circuits. *Neural Comp.* **10**:1321-1327.

Zhang LI, Tao, HW, Holt CE, Harris WA & Poo M-m (1998) A critical window for cooperation and competition among developing retinotectal synapses. *Nature* **395**:37-44.
